# Hierarchical Linear/Constant Time SLAM Using Particle Filters for Dense Maps

**Austin I. Eliazar**    **Ronald Parr**
Department of Computer Science
Duke University
Durham, NC 27708
{*eliazar,parr*}*@cs.duke.edu*

## Abstract

We present an improvement to the DP-SLAM algorithm for simultaneous localization and mapping (SLAM) that maintains multiple hypotheses about densely populated maps (one full map per particle in a particle filter) in time that is linear in all significant algorithm parameters and takes constant (amortized) time per iteration. This means that the asymptotic complexity of the algorithm is no greater than that of a pure localization algorithm using a single map and the same number of particles. We also present a hierarchical extension of DP-SLAM that uses a two level particle filter which models drift in the particle filtering process itself. The hierarchical approach enables recovery from the inevitable drift that results from using a finite number of particles in a particle filter and permits the use of DP-SLAM in more challenging domains, while maintaining linear time asymptotic complexity.

## 1 Introduction

The ability to construct and use a map of the environment is a critical enabling technology for many important applications, such as search and rescue or extraterrestrial exploration. Probabilistic approaches have proved successful at addressing the basic problem of localization using particle filters [6]. Expectation Maximization (EM) has been used successfully to address the problem of mapping [1] and Kalman filters [2, 10] have shown promise on the combined problem of simultaneous localization and mapping (SLAM).

SLAM algorithms ought to produce accurate maps with bounded resource consumption per sensor sweep. To the extent that it is possible, it is desirable to avoid explicit map correcting actions, which are computationally intensive and would be symptomatic of accumulating error in the map. One family of approaches to SLAM assumes relatively sparse, relatively unambiguous landmarks and builds a Kalman filter over landmark positions [2, 9, 10] . Other approaches assume dense sensor data which individually are not very distinctive, such as those available from a laser range finder [7, 8]. An advantage of the latter group is that they are capable of producing detailed maps that can be used for path planning.

In earlier work, we presented an algorithm called DP-SLAM [4], which produced extremely accurate, densely populated maps by maintaining a joint distribution over robot maps and poses using a particle filter. DP-SLAM uses novel data structures that exploit shared structure between maps, permitting efficient use of many joint map/pose particles.

This gives DP-SLAM the ability to resolve map ambiguities automatically, as a natural part of the particle filtering process, effectively obviating the explicit loop closing phase needed for other approaches [7, 12].

A known limitation of particle filters is that they can require a very large number of particles to track systems with diffuse posterior distributions. This limitation strongly affected earlier versions of DP-SLAM, which had a worst-case run time that scaled quadratically with the number of particles. In this paper, we present a significant improvement to DP-SLAM which reduces the run time to linear in the number of particles, giving multiple map hypothesis SLAM the same asymptotic complexity per particle as localization with a single map. The new algorithm also has a more straightforward analysis and implementation.

Unfortunately, even with linear time complexity, there exist domains which require infeasibly large numbers of particles for accurate mapping. The cumulative effect of very small errors (resulting from sampling or discretization) can cause drift. To address the issue of drift in a direct and principled manner, we propose a hierarchical particle filter method which can specifically model and recover from small amounts of drift, while maintaining particle diversity longer than in typical particle filters. The combined result is an algorithm that can produce extraordinarily detailed maps of large domains at close to real time speeds.

## 2   Linear Time Algorithm

A DP-SLAM *ancestry tree* contains all of the current particles as leaves. The parent of a given node represents the particle of the previous iteration from which that particle was resampled. An ancestry tree is *minimal* if the following two properties hold:

1. A node is a leaf node if and only if it corresponds to a current generation particle.
2. All interior nodes have at least two children.

The first property is ensured by simply removing particles that are not resampled from the ancestry tree. The second property is ensured by merging parents with only-child nodes. It is easy to see that for a particle filter with $P$ particles, the corresponding minimal ancestry tree will have a branching factor of at least two and depth of no more than $O(P)$.

The complexity of maintaining a minimal ancestry tree will depend upon the manner in which observations, and thus maps, are associated with nodes in the tree. DP-SLAM distributes this information in the following manner: All map updates for all nodes in the ancestry tree are stored in a single global grid, while each node in the ancestry tree also maintains a list of all grid squares updated by that node. The information contained in these two data structures is integrated for efficient access at each cycle of the particle filter through a new data structure called an *map cache*.

### 2.1   Core Data Structures

The DP-SLAM map is a global occupancy grid-like array. Each grid cell contains an *observation vector* with one entry for each ancestry tree node that has made an observation of the grid cell. Each vector entry is an *observation node* containing the following fields:

**opacity** a data structure storing sufficient statistics for the current estimate of the opacity of the grid cell to the laser range finder. See Eliazar and Parr [4] for details.

**parent** a pointer to a parent observation node for which this node is an update. (If an ancestor of a current particle has seen this square already, then the opacity value for this square is considered an update to the previous value stored by the ancestor. However, both the update and the original observation are stored, since it may not be the case that all successors of the ancestor have made updates to this square.)

**anode** a pointer to the ancestry tree node associated with the current opacity estimate.

In previous versions of DP-SLAM, this information was stored using a balanced tree. This added significant overhead to the algorithm, both conceptual and computational, and is no longer required in the current version.

The DP-SLAM *ancestry tree* is a basic tree data structure with pointers to parents and children. Each node in the ancestry tree also contains an **onodes** vector, which contains pointers to observation nodes in the grid cells updated by the ancestry tree node.

## 2.2 Map cache

The main sacrifice that was made when originally designing DP-SLAM was that map accesses no longer took constant time, due to the need to search the observation vector at a given grid square. The map cache provides a way of returning to this constant time access, by reconstructing a separate local map which is consistent with the history of map updates for each particle. Each local map is only as large as the area currently observed, and therefore is of a manageable size.

For a localization procedure using $P$ particles and observing an area of $A$ grid squares, there is a total of $O(AP)$ map accesses. For the constant time accesses provided by the map cache to be useful, the time complexity to build the map cache needs to be $O(AP)$. This result can be achieved by constructing the cache in two passes.

The first pass is to iterate over all grid squares in the global map which could be within sensor range of the robot. For each of these grid squares, the observation vector stores all observations made of that grid square by any particle. This vector is traversed, and for each observation, we update the corresponding local map with a pointer back to the corresponding observation node. This creates a set of partial local maps that store pointers to map updates, but no inherited map information. Since the size of the observation vector can be no greater than the size of the ancestry tree, which has $O(P)$ nodes, the first pass takes $O(P)$ time per grid square.

In the second pass we fill holes in the local maps by propagating inherited map information. The entire ancestry tree is traced, depth first, and the local map is checked for each ancestor node encountered. If the local map for the current ancestor node was not filled during the first pass, then the hole is patched by inheritance from the ancestor node's parent. This will fill any gaps in the local maps for grid squares that have been seen by any current particle. As this pass is directly based on the size of the ancestry tree, it is also $O(P)$ per grid square. Therefore, the total complexity of building the map cache is $O(AP)$.

For each particle, the algorithm constructs a grid of pointers to observation nodes. This provides constant time access to the opacity values consistent with each particle's map. Localization now becomes trivial with this representation: Laser scans are traced through the corresponding local map, and the necessary opacity values are extracted via the pointers. With the constant time accesses afforded by the local maps, the total localization cost in DP-SLAM is now $O(AP)$.

## 2.3 Updates and Deletions

When the observations associated with a new particle's sensor sweep are integrated into the map, two basic steps are performed. First, a new observation is added to the observation vector of each grid square which was visited by the particle's laser casts. Next, a pointer to each new observation is added to this particle's **onodes** vector. The cost of this operation is obviously no more than that of localization.

There are two situations which require deleting nodes from the ancestry tree. The first is

the simple case of removing a node from which the particle filter has not resampled. Each ancestor node maintains a vector of pointers to all observations attributed to it. Therefore, these entries can be removed from the observation vectors in the global grid in constant time. Since there can be no more deletions than there are updates, this process has an amortized cost of $O(AP)$.

The second case for deleting a node occurs when a node in the ancestry tree which has an only child is merged with that child. This involves replacing the opacity value for the parent with that of the child, and then removing that child's entry from the associated grid cell's observation vector. Therefore, this process is identical to the first case, except that each removal of an entry from the global map is preceded by a single update to the same grid square. Since the observation vector at each grid square is not ordered, additions to the vector can be done in constant time, and does not change the complexity from $O(AP)$.

## 3  Drift

A significant problem faced by current SLAM algorithms is that of drift. Small errors can accumulate over several iterations, and while the resulting map may seem locally consistent, there could be large total errors, which become apparent after the robot closes a large loop. In theory, drift can be avoided by some algorithms in situations where strong linear Gaussian assumptions hold [10]. In practice, it is hard to avoid drift, either as a consequence of violated assumptions or as a consequence of particle filtering. The best algorithms can only extend the distance that the robot travels before experiencing drift. Errors come from (at least) three sources: insufficient particle coverage, coarse precision, and resampling itself (particle depletion).

The first problem is a well known issue with particle filters. Given a finite number of particles, there will be unsampled gaps in the particle coverage of the state space and the proximity to the true state can be as coarse as the size of these gaps. This is exacerbated by the fact that particle filters are often applied to high dimensional state spaces with Gaussian noise, making it impossible to cover unlikely (but still possible) events in the tails of distribution with high particle density. The second issue is coarse precision. This can occur as a result of explicit discretization through an occupancy grid, or implicit discretization through the use of a sensor with finite precision. Coarse precision can make minor perturbations in the state appear identical from the perspective of the sensors and the particle weights. Finally, resampling itself can lead to drift by shifting a finite population of particles away from low probability regions of the state space. While this behavior of a particle filter is typically viewed as a desirable reallocation of computational resources, it can shift particles away from the true state in some cases.

The net effect of these errors can be the gradual accumulation of small errors resulting from failure to sample, differentiate, or remember a state vector that is sufficiently close to the true state. In practice, we have found that there exist large domains where high precision mapping is essentially impossible with any reasonable number of particles.

## 4  Hierarchical SLAM

In the first part of the paper, we presented an approach to SLAM that reduced the asymptotic complexity per particle to that of pure localization. This is likely as low as can reasonably be expected and should allow the use of large numbers of particles for mapping. However, the discussion of drift in the previous section underscores that the ability to use large numbers of particles may not be sufficient, and we would like techniques that delay the onset of drift as long as possible. We therefore propose a hierarchical approach to SLAM that is capable of recognizing, representing, and recovering from drift.

The basic idea is that the main sources of drift can be modeled as the cumulative effect of a sequence of random events. Through experimentation, we can quantify the expected amount of drift over a certain distance for a given algorithm, much in the same way that we create a probabilistic motion model for the noise in the robot's odometry. Since the total drift over a trajectory is assumed to be a summation of many small, largely independent sources of error, it will be close to a Gaussian distribution.

If we view the act of completing a small map *segment* as a random process with noise, we can then apply a higher level filter to the output of the map segment process in an attempt to track the underlying state more accurately. There are two benefits to this approach. First, it explicitly models and permits the correction of drift. Second, the coarser time granularity of the high level process implies fewer resampling steps and fewer opportunities for particle depletion. Thus, if we can model how much drift is expected to occur over a small section of the robot's trajectory, we can maintain this extra uncertainty longer, and resolve inaccuracies or ambiguities in the map in a natural fashion.

There are some special properties of the SLAM problem that make it particularly well suited to this approach. In the full generality of an arbitrary tracking problem, one should view drift as a problem that affects entire trajectories through state space and the complete belief state at any time. Sampling the space of drifts would then require sampling perturbations to the entire state vector. In this fully general case, the benefit of the hierarchical view would be unclear, as the end result would be quite similar to adding additional noise to the low level process. In SLAM, we can make two assumptions that simplify things. The first is that the robot state vector is highly correlated with the remaining state variables, and the second is that we have access to a low level mapping procedure with moderate accuracy and local consistency. Under these assumptions, the the effects of drift on low level maps can be accurately approximated by perturbations to the endpoints of the robot trajectory used to construct a low level map. By sampling drift only at endpoints, we will fail to sample some of the internal structure that is possible in drifts, e.g., we will fail to distinguish between a linear drift and a spiral pattern with the same endpoints. However, the existence of significant, complicated drift patterns within a map segment would violate our assumption of moderate accuracy and local consistency within our low level mapper.

To achieve a hierarchical approach to SLAM, we use a standard SLAM algorithm using a small portion of the robot's trajectory as input for the low level mapping process. The output is not only a distribution over maps, but also a distribution over robot trajectories. We can treat the distribution over trajectories as a distribution over motions in the higher level SLAM process, to which additional noise from drift is added. This allows us to use the output from each of our small mapping efforts as the input for a new SLAM process, working at a much higher level of time granularity.

For the high level SLAM process, we need to be careful to avoid double counting evidence. Each low level mapping process runs as an independent process intialized with an empty map. The distribution over trajectories returned by the low level mapping process incorporates the effects of the observations used by the low level mapper. To avoid double counting, the high level SLAM process can only weigh the match between the new observations and the existing high level maps. In other words, *all* of the observations for a single high level motion step (single low level trajectory) must be evaluated against the high level map, before any of those observations are used to update the map. We summarize the high level SLAM loop for each high level particle as follows:

1. Sample a high level SLAM state (high level map and robot state).

2. Perturb the sampled robot state by adding random drift.

3. Sample a low level trajectory from the distribution over trajectories returned by the low level SLAM process.

4. Compute a high level weight by evaluating the trajectory and robot observations against the

sampled high level map, starting from the perturbed robot state.

5. Update the high level map based upon the new observations.

In practice this can give a much greater improvement in accuracy over simply doubling the resources allocated to a single level SLAM algorithm because the high level is able to model and recover from errors much longer than would be otherwise possible with only a single particle filter. In our implementation we used DP-SLAM at both levels of the hierarchy to ensure a total computational complexity of $O(AP)$. However, there is reason to believe that this approach could be applied to any other sampling-based SLAM method just as effectively. We also implemented this idea with only one level of hierarchy, but multiple levels could provide additional robustness. We felt that the size of the domains on which we tested did not warrant any further levels.

## 5 Implementation and Empirical Results

Our description of the algorithm and complexity analysis assumes constant time updates to the vectors storing information in the core DP-SLAM data structures. This can be achieved in a straightforward manner using doubly linked lists, but a somewhat more complicated implementation using adjustable arrays is dramatically more efficient in practice. A careful implementation can also avoid caching maps for interior nodes of the ancestry tree.

As with previous versions of DP-SLAM, we generate many more particles than we keep at each iteration. Evaluating a particle requires line tracing 181 laser casts. However, many particles will have significantly lower probability than others and this can be discovered before they are fully evaluated. Using a technique we call *particle culling* we use partial scan information to identify and discard lower probability particles before they are evaluated fully. In practice, this leads to large reduction in the number of laser casts that are fully traced through the grid. Typically, less than one tenth of the particles generated are resampled.

For a complex algorithm like DP-SLAM, asymptotic analysis may not always give a complete picture of real world performance. Therefore, we provide a comparison of actual run times for each method on three different data logs. The particle counts provided are the minimum number of particles needed (at each level) to produce high-quality maps reliably. The improved run time for the linear algorithm also reflects the benefits of some improvements in our culling technique and a cleaner implementation permitted by the linear time algorithm. The quadratic code is simply too slow to run on the Wean Hall data. Log files for these runs are available from the DP-SLAM web page: `http://www.cs.duke.edu/~parr/dpslam/`. The results show a significant practical advantage for the linear code, and vast improvement, both in terms of time and number of particles, for the hierarchical implementation.

| | Quadratic | | Linear | | Hierarchical | |
|---|---|---|---|---|---|---|
| Log | Particles | Minutes | Particles | Minutes | Particles (high/low) | Minutes |
| loop5 | 1500 | 55 | 1500 | 14 | 200/250 | 12 |
| loop25 | 11000 | 1345 | 11000 | 690 | 2000/3000 | 289 |
| Wean Hall | 120000 | N/A | 120000 | 2535 | 2000/3000 | 293 |

Finally, in Figure 1 we include sample output from the hierarchical mapper on the Wean Hall data shown in our table. In this domain, the robot travels approximately 220m before returning to its starting position. Each low level SLAM process was run for 75 time steps, with an average motion of 12cm for each time step. The nonhierarchical approach can produce a very similar result, but requires at least 120,000 particles to do so reliably. (Smaller numbers of particles produced maps with noticeable drifts and errors.) This extreme difference in particle counts and computation time demonstrates the great improvement that can be realized with the hierarchical approach. (The Wean Hall dataset has been mapped

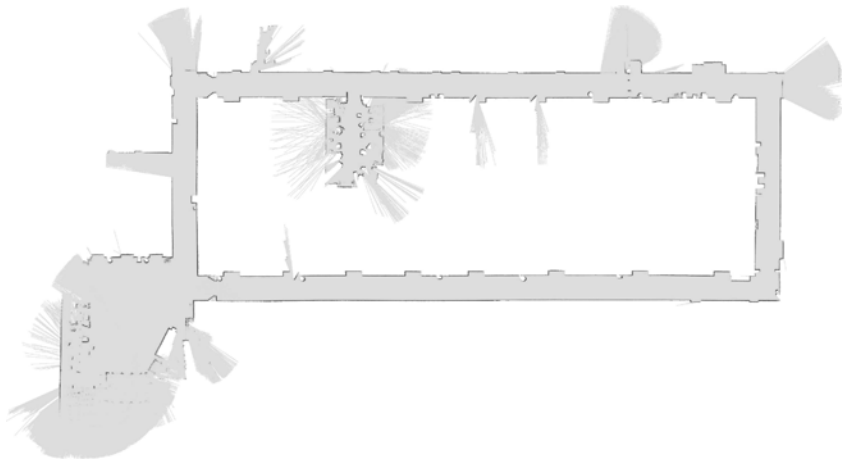

Figure 1: CMU's Wean Hall at 4cm resolution, using hierarchical SLAM. Please zoom in on the map using a software viewer to appreciate some of the fine detail.

successfully before at low resolution using a non-hierarchical approach with run time per iteration that grows with the number of iterations [8].)

## 6  Related Work

Other methods have attempted to preserve uncertainty for longer numbers of time steps. One approach seeks to delay the resampling step for several iterations, so as to address the total noise in a certain number of steps as one Gaussian with a larger variance [8]. In general, look-ahead methods can "peek" at future observations to use the information from later time steps to influence samples at a previous time step [3]. The HYMM approach[11] combines different types of maps. Another way to interpret hierarchical SLAM is in terms of a hierarchical hidden Markov model framework [5]. In a hierarchical HMM, each node in the HMM has the potential to invoke sub-HMMs to produce a series of observations. The main difference is that in hierarchical HMMs, there is assumed to be a single process that can be represented in different ways. In our hierarchical SLAM approach, only the lowest level models a physical process, while higher levels model the errors in lower levels.

## 7  Conclusions and Future Research

We have presented a SLAM algorithm which is the culmination of our efforts to make multiple hypothesis mapping practical for densely populated maps. Our first algorithmic accomplishment is to show that this requires no more effort, asymptotically, than pure localization using a particle filter. However, for mapping, the number of particles needed can be large and can still grow to be unmanageable for large domains due to drift. We therefore developed a method to improve the accuracy achieveable with a reasonable number of particles. This is accomplished through the use of a hierarchical particle filter. By allowing an additional level of sampling on top of a series of small particle filters, we can successfully maintain the necessary uncertainty to produce very accurate maps. This is due to the explicit modeling of the drift, a key process which differentiates this approach from previous attempts to preserve uncertainty in particle filters.

The hierarchical approach to SLAM has been shown to be very useful in improving DP-SLAM performance. This would lead us to believe that similar improvements could also be

realized in applying this to other sampling based SLAM methods. SLAM is perhaps not the only viable application for hierarchical framework for particle filters. However, one of the key aspects of SLAM is that the drift can easily be represented by a very low dimensional descriptor. Other particle filter applications which have drift that must be modeled in many more dimensions could benefit much less from this hierarchical approach.

The work of Hahnel et al. [8] has made progress in increasing efficiency and reducing drift by using scan matching rather than pure sampling from a noisy proposal distribution. Since much of the computation time used by DP-SLAM is spent evaluating bad particles, a combination of DP-SLAM with scan matching could yield significant practical speedups.

## Acknowledgments

This research was supported by SAIC, the Sloan foundation, and the NSF. The Wean Hall data were gracriously provided by Dirk Hahnel and Dieter Fox.

## References

[1] W. Burgard, D. Fox, H. Jans, C. Matenar, and S. Thrun. Sonar-based mapping with mobile robots using EM. In *Proc. of the International Conference on Machine Learning*, 1999.

[2] P. Cheeseman, P. Smith, and M. Self. Estimating uncertain spatial relationships in robotics. In *Autonomous Robot Vehicles*, pages 167–193. Springer-Verlag, 1990.

[3] N. de Freitas, R. Dearden, F. Hutter, R. Morales-Menendez, J. Mutch, and D. Poole. Diagnosis by a waiter and a Mars explorer. In *IEEE Special Issue on Sequential State Estimation*, pages 455–468, 2003.

[4] A. Eliazar and R. Parr. DP-SLAM 2.0. In *IEEE International Conference on Robotics and Automation (ICRA)*, 2004.

[5] Shai Fine, Yoram Singer, and Naftali Tishby. The hierarchical hidden markov model: Analysis and applications. *Machine Learning*, 32(1):41–62, 1998.

[6] Dieter Fox, Wolfram Burgard, Frank Dellaert, and Sebastian Thrun. Monte carlo localization: Efficient position estimation for mobile robots. In *AAAI-99*, 1999.

[7] J. Gutmann and K. Konolige. Incremental mapping of large cyclic environments. In *IEEE International Symposium on Computational Intelligence in Robotics and Automation (ICRA)*, pages 318–325, 2000.

[8] Dirk Hahnel, Wolfram Burgard, Dieter Fox, and Sebastian Thrun. An efficient fastslam algorithm for generating maps of large-scale cyclic environments from raw laser range measurements. In *Proceedings of the International Conference on Intelligent Robots and Systems*, 2003.

[9] John H. Leonard, , and Hugh F. Durrant-Whyte. Mobile robot localization by tracking geometric beacons. In *IEEE Transactions on Robotics and Automation*, pages 376–382. IEEE, June 1991.

[10] M. Montemerlo, S. Thrun, D. Koller, and B. Wegbreit. FastSLAM 2.0: An improved particle filtering algorithm for simultaneous localization and mapping that provably converges. In *IJCAI-03*, Morgan Kaufmann, 2003. 1151–1156.

[11] J. Nieto, J. Guivant, and E. Nebot. The HYbrid Metric Maps (HYMMS): A novel map representation for denseSLAM. In *IEEE International Conference on Robotics and Automation (ICRA)*, 2004.

[12] S. Thrun. A probabilistic online mapping algorithm for teams of mobile robots. *International Journal of Robotics Research*, 20(5):335–363, 2001.
